# $H^\infty$ Optimal Training Algorithms and their Relation to Backpropagation

**Babak Hassibi***
Information Systems Laboratory
Stanford University
Stanford, CA 94305

**Thomas Kailath**
Information Systems Laboratory
Stanford University
Stanford, CA 94305

## Abstract

We derive *global* $H^\infty$ optimal training algorithms for neural networks. These algorithms guarantee the smallest possible prediction error energy over *all* possible disturbances of fixed energy, and are therefore robust with respect to model uncertainties and lack of statistical information on the exogenous signals. The ensuing estimators are infinite-dimensional, in the sense that updating the weight vector estimate requires knowledge of all previous weight esimates. A certain finite-dimensional approximation to these estimators is the backpropagation algorithm. This explains the *local $H^\infty$* optimality of backpropagation that has been previously demonstrated.

## 1 Introduction

Classical methods in estimation theory (such as maximum-likelihood, maximum entropy and least-squares) require a priori knowledge of the statistical properties of the exogenous signals. In some applications, however, one is faced with model uncertainties and lack of statistical information, which has led to an increasing interest in minimax estimation (see *e.g.*, Zames 1981, Khargonekar and Nagpal 1991, and the references therein) with the belief that the resulting so-called $H^\infty$ algorithms will be more robust and less sensitive to parameter variations.

In (Hassibi, Sayed and Kailath, 1994) we have shown that LMS (Widrow and Hoff, 1960) and backpropagation (Rumelhart and Mclelland, 1986), the currently most widely used adaptive algorithms that have long been considered to be approximate $H^2$ (or least-squares) solutions, are indeed $H^\infty$ optimal and locally $H^\infty$ optimal algorithms, respectively. This, in our view, connects earlier work in learning theory to more recent ideas in robust estimation, and explains why LMS and backpropagation have found wide applicability in such a diverse range of problems.

The local $H^\infty$ optimality of backpropagation implies that backpropagation minimizes the energy gain from the disturbances to the prediction errors, only if the initial condition is close enough to the true weight vector and if the disturbances are small enough. In this paper we derive global $H^\infty$ optimal estimators that minimize the energy gain from the disturbances to the prediction errors for *all* initial conditions and disturbances. The resulting estimator (given by Theorem 1) has growing memory, which we refer to as being infinite-dimensional, since updating the weight vector estimate requires knowledge of all previous weight estimates. When the underlying model is linear, we show that this infinite-dimensional estimator reduces to the finite-dimensional LMS filter. When the underlying model is nonlinear, the infinite-dimensionality of the estimator may prohibit its practical applicablity, and one needs to construct finite-dimensional approximations to this estimator. We consider two such approximations here: one yields the backpropagation algorithm, and the other is a second-order algorithm based on the Newton-Raphson iteration. There are, no doubt, a wide variety of other approximations which should be worthy of further scrutiny.

## 2   Robust Estimation

In estimation problems one assumes a certain model (say an FIR filter in adaptive filtering, or a neural network), observes a corrupted version of the output of this model, and wants to estimate the parameters associated with this model (say the weights of the FIR filter or neural network). Most estimation algorithms make some *assumption* about the nature of the disturbances, and then proceed to estimate the parameters using some optimality criterion. To be more specific, we shall consider the following two cases.

### 2.1   The Linear Case

Suppose that we observe an output sequence $\{d_i\}$ that obeys the following linear model

$$d_i = x_i^T w + v_i, \tag{1}$$

where $x_i^T = [\begin{array}{cccc} x_{i1} & x_{i2} & \cdots & x_{in} \end{array}]$ is a known input vector, $w$ is the unknown weight vector that we intend to estimate, and $\{v_i\}$ is an unknown disturbance sequence. Let $w_i = \mathcal{F}(d_0, d_1, \ldots, d_i)$ denote the estimate of the weight vector given the inputs $\{x_j\}$ and the outputs $\{d_j\}$ from time 0 up to and including time $i$. The most widely used estimate $w_i$, is one that satisfies the following $H^2$ criterion

$$\min_w \left[ \mu^{-1} |w - w_{-1}|^2 + \sum_{j=0}^{i} |d_j - x_j^T w|^2 \right], \tag{2}$$

where $\mu$ is a positive constant that reflects a priori knowledge as to how close $w$ is to the initial estimate $w_{-1}$. The *exact* solution to the above criterion is given by the RLS algorithm (Haykin, 1991):

$$w_i = w_{i-1} + k_{p,i}(d_i - x_i^T w), \qquad w_{-1} \tag{3}$$

where $w_{-1}$ denotes the initial value,

$$k_{p,i} = \frac{P_i x_i}{1 + x_i^T P_i x_i}$$

and

$$P_{i+1} = P_i - \frac{P_i x_i x_i^T P_i}{1 + x_i^t P_i x_i}, \qquad P_0 = \mu I.$$

If we assume that in the model (1) the $w - w_{-1}$ and $\{v_i\}$ are zero-mean independent Gaussian random variables with variances $\mu I$ and 1, respectively, then the cost function in (2) is simply the associated log-likelihood function. Thus the estimate given by minimizing (2) will be the maximum-likelihood estimate of the weight vector $w$. In particular, it can be shown that under these assumptions, RLS minimizes the expected prediction energy

$$E \parallel e \parallel_i^2 = E \sum_{j=0}^{i} |x_j^T w - x_j^T w_{j-1}|^2.$$

Note that the LMS algorithm is an approximation to RLS where $k_{p,i}$ is replaced by $\mu x_i$, so that the estimates are updated along the direction of the instantaneous gradient of (2):

$$w_i = w_{i-1} + \mu x_i(d_i - x_i^T w), \qquad w_{-1}. \qquad (4)$$

## 2.2 The Nonlinear Case

Suppose now that we observe an output sequence $\{d_i\}$ that obeys the following nonlinear model

$$d_i = g_i(w) + v_i, \qquad (5)$$

where $g_i(.)$ is a known *nonlinear* function (with bounded first and second order derivatives), $w$ is the unknown weight vector we intend to estimate, and $\{v_i\}$ is an unknown disturbance sequence. In a neural network context the index $i$ in $g_i(.)$ will correspond to the nonlinear function that maps the weight vector to the output when the $i$th input pattern is presented, i.e., $g_i(w) = g(x_i, w)$ where $x_i$ is the $i$th input pattern. As before we shall denote by $w_i = \mathcal{F}(d_0, \ldots, d_i)$ the estimate of the weight vector using measurements up to and including time $i$. The $H^2$ criterion for finding the estimate is

$$\min_w \left[ \mu^{-1} |w - w_{-1}|^2 + \sum_{j=0}^{i} |d_j - g_j(w)|^2 \right]. \qquad (6)$$

As in the linear case, if we assume that in the model (5) the disturbances $w - w_{-1}$ and $\{v_i\}$ are zero-mean independent Gaussian random variables with variances $\mu I$ and 1, respectively, then the cost function in (6) is the log-likelihood function and the weight vector that minimizes it is the maximum-likelihood estimate. However, contrary to the linear case, the solution to (6) will *not*, in general, minimize the expected prediction error energy.

In the nonlinear case exact solutions to (6) do not exist, and the backpropagation algorithm is a generalization of the LMS algorithm where once more the estimates are updated along the negative direction of the instantaneous gradient of the log-likelihood function:

$$w_i = w_{i-1} + \mu \frac{\partial g_i}{\partial w}(w_{i-1})(d_i - g_i(w_i)), \qquad w_{-1}. \qquad (7)$$

Generalizations of the RLS algorithm to the nonlinear setting are the second order Gauss-Newton methods.

## 2.3   The Question of Robustness

In view of the above discussion we have seen that $H^2$-optimal estimation strategies (see (2) and (6)) are maximum-likelihood and minimize the expected prediction error energy (in the linear case), if we assume that the disturbances are zero-mean independent Gaussian random variables. However, the question that begs itself is what the performance of such estimators will be if the assumptions on the disturbances are violated, or if there are modelling errors in our model so that the disturbances must include the modelling errors? In other words

- is it possible that *small* disturbances and modelling errors may lead to *large* estimation errors?

Obviously a nonrobust algorithm would be one for which the above is true, and a robust algorithm would be one for which small disturbances lead to small estimation errors. (For example in the adaptive filtering problem, where we assumed an FIR model, the *true* model may be IIR, but we neglect the tail of the filter since its components are small. However, unless one uses a robust estimation algorithm, it is conceivable that this small modelling error may result in large estimation errors.)

The problem of robust estimation is thus an important one, and is worthy of study in its own right. Rather surprisingly, it had not received much attention until quite recently. The $H^\infty$ criterion has been introduced (Zames, 1981) as a means of studying such questions in the contexts of estimation and control. This is the subject of the next section.

## 3   The $H^\infty$ Problem

The $H^\infty$ estimation formulation is an attempt to address the robustness question raised in the previous section. The idea is to come up with estimators that minimize (or in the suboptimal case bound) the maximum energy gain from the disturbances to the estimation errors. This will guarantee that if the disturbances are small (in energy) then the estimation errors will be as small as possible (in energy), *no matter what the disturbances are*. In other words the maximum energy gain is minimized over *all possible* disturbances. The robustness of the $H^\infty$ estimators arises from this fact. Since they make no assumption about the disturbances, they have to accomodate for all conceivable disturbances, and are thus over-conservative.

We once more assume that we observe an output sequence $\{d_i\}$ that obeys the following nonlinear model

$$d_i = g_i(w) + v_i, \tag{8}$$

where $g_i(.)$ is a known *nonlinear* function, $w$ is an unknown weight vector, and $\{v_i\}$ is an unknown disturbance sequence that includes noise and/or modelling errors. Recall that in a neural network context $g_i(w) = g(x_i, w)$, where $x_i$ is the $i$th input pattern. As before we shall denote by $w_i = \mathcal{F}(d_0, \ldots, d_i)$ the estimate of the weight vector using measurements up to and including time $i$, and the prediction error by

$$e_i = g_i(w) - g_i(w_{i-1}).$$

The optimal $H^\infty$ estimation problem may now be stated as follows.

**Problem 1 (Optimal $H^\infty$ Estimation Problem)** *Find an $H^\infty$-optimal estimation strategy $w_i = \mathcal{F}(d_0, d_1, \ldots, d_i)$ that minimizes the maximum energy gain from the disturbances $w - w_{-1}$ and $\{v_i\}$ to the prediction errors $\{e_i = g_i(w) - g_i(w_{i-1})\}$, and obtain the resulting*

$$\gamma_{opt}^2 = \inf_{\mathcal{F}} \sup_{w,v \in h_2} \frac{\|e\|_i^2}{\mu^{-1}|w - w_{-1}|^2 + \|v\|_i^2} = \inf_{\mathcal{F}} \sup_{w,v \in h_2} \frac{\sum_{j=0}^i |g_j(w) - g_j(w_{j-1})|^2}{\mu^{-1}|w - w_{-1}|^2 + \sum_{j=0}^i |v_j|^2} \tag{9}$$

*where $\mu > 0$ reflects a priori knowledge of how close $w$ is to the initial estimate $w_{-1}$, and where $h_2$ is the space of all causal square-summable sequences. $\gamma_{opt}$ is the so-called minimum $H^\infty$ norm.*

Note that the infimum in (9) is taken over all *causal* estimators $\mathcal{F}$. Although the $H^\infty$ estimation problem has been solved in the linear case, to date there does not exist a satisfactory solution for the nonlinear case, and indeed the class of nonlinear functions $g_i(.)$ for which the above problem has a solution is not known (Ball and Helton, 1992).

We have, however, been able to solve Problem 1 in the case where the $g_i(.)$ are bounded functions with bounded first and second order derivatives. These conditions are of course satisfied by multi-layer neural networks with sigmoidal elements. The result is stated below, where we call the column vectors $\{x_i\}$ *exciting* if $\lim_{i \to \infty} \sum_{j=0}^{i} x_j^t x_j = \infty$.

**Theorem 1 ($H^\infty$ Optimal Algorithm)** *Consider the model (8) where the $g_i(.)$ are bounded and have bounded first and second order derivatives, and suppose we wish to minimize the maximum energy gain from the unknowns $w - w_{-1}$ and $\{v_i\}$ to the prediction errors $\{e_i\}$. If*

$$0 < \mu < \inf_i \inf_w \frac{1}{|\frac{\partial g_i}{\partial w}(w)|^2},$$ 
(10)

*and the $\{\frac{\partial g_i}{\partial w}(w)\}$ are exciting, then the minimum $H^\infty$ norm is*

$$\gamma_{opt} = 1.$$

*In this case an optimal $H^\infty$ estimator is given by the following sequence of nonlinear equations*

$$
\begin{cases}
\frac{1}{\mu} w_0 &= (d_0 - g_0(w_{-1}))\frac{\partial g_0}{\partial w}(w_0) \\
\\
\frac{1}{\mu} w_1 &= (d_0 - g_0(w_{-1}))\frac{\partial g_0}{\partial w}(w_1) + (d_1 - g_1(w_0))\frac{\partial g_1}{\partial w}(w_1) \\
\vdots & \vdots \\
\frac{1}{\mu} w_i &= (d_0 - g_0(w_{-1}))\frac{\partial g_0}{\partial w}(w_i) + (d_1 - g_1(w_0))\frac{\partial g_1}{\partial w}(w_i) + \\
& \qquad \ldots + (d_i - g_i(w_{i-1}))\frac{\partial g_i}{\partial w}(w_i) \\
\vdots & \vdots
\end{cases}
$$
(11)

**Remarks:**

(i) The fact that $\hat{g}_i(w) = g_i(w_{i-1})$ implies that the output prediction has the same *structure* as our model (*i.e.* that there exists a weight vector estimate $w_{i-1}$ that yields the desired output prediction).

(ii) Theorem 1 states that $\gamma_{opt} = 1$. While it is not intuitively difficult to convince oneself that $\gamma_{opt}$ cannot be less than one (simply choose the disturbances $v_i$ so that $v_i = e_i$, whereby the ratio in (9) can be made arbitrarily close to one), the surprising fact is that $\gamma_{opt}$ is one. What this means is that the estimator of Theorem 1 guarantees that the energy of the prediction errors will never exceed the energy of the disturbances. This is of course not true of other estimators.

(iii) Theorem 1 gives an upper bound on the quantity $\mu$ that guarantees $\gamma_{opt} = 1$. As we shall see below, the $\mu$ of Theorem 1 is a generalization of the learning rate $\mu$ of the LMS and backpropagation algorithms (see (4) and (7)), and this is in accordance with the well-known fact that LMS and backpropagation behave poorly if the learning rate is chosen too large.

(iv) In view of Theorem 1, to obtain the estimate $w_i$ we need to solve a nonlinear equation that involves all previous estimates $w_0, \ldots, w_{i-1}$. This means that the estimator (11) is infinite-dimensional. Although this may prohibit practical applications of this algorithm, it will be very useful to study special cases under which the estimator becomes finite-dimensional, or to find finite-dimensional approximations for (11). This will be done in the next section.

## 4  Special Cases

### 4.1  The Linear Case

In the linear case the model we consider has

$$g_i(w) = x_i^T w,$$

so that $\frac{\partial g_i}{\partial w}(w) = x_i$. Although the linear function $g_i(w) = x_i^T w$ does not satisfy the boundedness condition of Theorem 1, let us investigate the consequence of applying algorithm (11) to this case. Thus the $(i+1)$th equation in (11) becomes

$$\frac{1}{\mu} w_i = (d_0 - x_0^T w_{-1})x_0 + (d_1 - x_1^T w_0)x_1 + \ldots + (d_{i-1} - x_{i-1}^T w_{i-2})x_{i-1} + (d_i - x_i^T w_{i-1})x_i.$$

But from the $i$th equation

$$\frac{1}{\mu} w_{i-1} = (d_0 - x_0^T w_{-1})x_0 + (d_1 - x_1^T w_0)x_1 + \ldots + (d_{i-1} - x_{i-1}^T w_{i-2})x_{i-1}$$

so that

$$\frac{1}{\mu} w_i = \frac{1}{\mu} w_{i-1} + (d_i - x_i^T w_{i-1})x_i, \tag{12}$$

which is the LMS algorithm (4). Thus in the linear case the estimator of Theorem 1 specializes to the LMS algorithm. This is expected since we have shown in (Hassibi et al., 1994) that the LMS algorithm is $H^\infty$ optimal. The result obtained there is as follows.

**Theorem 2 (LMS Algorithm)** *Consider the model (1), and suppose we wish to minimize the maximum energy gain from the unknowns $w - w_{-1}$ and $v_i$ to the prediction errors $e_i$. If the input vectors $x_i$ are exciting and*

$$0 < \mu < \inf_i \frac{1}{|x_i|^2} \tag{13}$$

*then the minimum $H^\infty$ norm is $\gamma_{opt} = 1$. In this case an optimal $H^\infty$ estimator is given by the LMS algorithm with learning rate $\mu$, viz.*

$$w_i = w_{i-1} + \mu x_i (d_i - x_i^T w_{i-1}) \quad , \qquad w_{-1} \tag{14}$$

Note that in the linear case the estimator is finite-dimensional since to find $w_i$ we only require knowledge of $w_{i-1}$.

### 4.2  Backpropagation

As mentioned at the end of Section 3, the $H^\infty$ optimal estimator of Theorem 1 is infinite-dimensional, in the sense that to obtain the estimate $w_i$ we need all previous estimates $w_0, \ldots, w_{i-1}$. We may obtain finite-dimensional approximations to the estimator (11) by constructing approximations to the nonlinear equations appearing

in (11). However, the resulting estimators will no longer be $H^\infty$ optimal in a global sense, but will only have local optimality.

The method that we shall use to obtain such approximate estimators is to assume that we have found the estimate $w_{i-1}$, and to use it as an initial guess to solve the $(i+1)$th equation in (11) whose solution is $w_i$. Depending on what algorithm we use to solve the $(i+1)$th equation with initial guess $w_{i-1}$, we shall get a different approximate estimator to (11).

To this end, suppose that we have solved the $i$th equation in (11) and have obtained $w_{i-1}$, i.e.

$$\frac{1}{\mu}w_{i-1} = (d_0 - g_0(w_{-1}))\frac{\partial g_0}{\partial w}(w_{i-1}) + \ldots + (d_{i-1} - g_{i-1}(w_{i-2}))\frac{\partial g_{i-1}}{\partial w}(w_{i-1}).$$

We now intend to solve the $(i+1)$th equation in (11) for $w_i$. Note that this equation is of the form $x = f(x)$ (where $x = w_i$). If we use one step of the *fixed-point iteration* method $x_{j+1} = f(x_j)$ with initial condition $x_0 = w_{i-1}$, we have

$$\frac{1}{\mu}w_i = \underbrace{(d_0 - g_0(w_{-1}))\frac{\partial g_0}{\partial w}(w_{i-1}) + \ldots + (d_{i-1} - g_{i-1}(w_{i-2}))\frac{\partial g_{i-1}}{\partial w}(w_{i-1})}_{\frac{1}{\mu}w_{i-1}} \quad (15)$$

$$+(d_i - g_i(w_{i-1}))\frac{\partial g_i}{\partial w}(w_{i-1}) \quad (16)$$

$$= \frac{1}{\mu}w_{i-1} + (d_i - g_i(w_{i-1}))\frac{\partial g_i}{\partial w}(w_{i-1}) \quad (17)$$

which is the backpropagation algorithm (7). Note that since we only use $w_{i-1}$ to compute $w_i$, backpropagation is a finite-dimensional approximation to the global $H^\infty$ optimal estimator (11). Due to its approximate nature, backpropagation has only *local $H^\infty$* optimality properties, as we have shown in (Hassibi et al., 1994). The result is stated below, where the column vectors $\{x_i\}$ are called *persistently exciting* if, $\lim_{i\to\infty}\frac{1}{i}\sum_{j=0}^{i} x_j x_j^t > \alpha I$, for some $\alpha > 0$.

**Theorem 3 (Local $H^\infty$ Optimality)** *Consider the model (8) and the backpropagation algorithm (7). Suppose that the $\frac{\partial g_i}{\partial w}(w_{i-1})$ are persistently exciting, and that (10) is satisfied. Then for each $\epsilon > 0$, there exist $\delta_1, \delta_2 > 0$ such that for all $|w - w_{-1}| < \delta_1$ and all $v \in h_2$ with $|v_i| < \delta_2$, we have*

$$\frac{\|e\|^2}{\mu^{-1}|w - w_{-1}|^2 + \|v\|^2} \leq 1 + \epsilon$$

Note that contrary to the global Theorem 1, backpropagation cannot achieve $\gamma_{opt} = 1$, and that it bounds the energy gain by $\sqrt{1 + \epsilon}$ only for *small enough* disturbances.

### 4.3 A Second-Order Algorithm

If instead of using one step of the fixed-point iteration to solve for $w_i$, as was done in Section 4.2, we use one step of the Newton-Raphson method with initial condition $w_{i-1}$, we obtain the following algorithm as an approximation to (11).

$$\begin{cases} w_i &= w_{i-1} + \mu(d_i - g_i(w_{i-1}))\Phi_i \frac{\partial g_i}{\partial w}(w_{i-1}), & w_{-1} \\ \Phi_i &= \Phi_{i-1} + \mu(d_{i-1} - g_{i-1}(w_{i-2}))\frac{\partial^2 g_{i-1}}{\partial w^2}(w_{i-2}), & \Phi_0 = I. \end{cases} \quad (18)$$

As before, (18) has only local optimality properties. However, since the Newton-Raphson method is less crude than the fixed-point iteration, it is expected to have

better local properties than backpropagation. The complexity of the algorithm is $O(n^2)$ per iteration which is, of course, higher than backpropagation which requires only $O(n)$ per iteration.

## 5 Conclusion

We have derived *global* $H^\infty$ optimal estimators for training neural networks. Such $H^\infty$ optimal algorithms will be most applicable in uncertain environments where there may be modelling errors, and where the statistics and/or distributions of the disturbances are not known (or are too expensive to obtain).

The resulting $H^\infty$ optimal algorithm of Theorem 1 is infinite-dimensional, so that computing the most recent weight vector estimate requires knowledge of all previous weight estimates. We considered two finite-dimensional approximations to this estimator (one of which was backpropagation) with the property that constructing the most recent weight estimate required only the immediately preceding weight estimate. However, the estimator of Theorem 1 has a very interesting structure that should allow for a wide variety of approximations, some of which may yield alternatives to the backpropagation algorithm. In particular, it would be interesting to study the possiblity of constructing estimators where updating the weight estimates requires more than one (but only finitely many) previous estimates.

The estimators constructed in this paper used prediction error as their criterion and should therefore have good generalization properties. It is also possible to construct similar estimators using filtered or smoothing error as the criterion, though this was not done due to lack of space.

### Acknowledgements

This work was supported in part by the Air Force Office of Scientific Research, Air Force Systems Command under Contract AFOSR91-0060 and by the Army Research Office under contract DAAL03-89-K-0109.

## Footnotes

*Contact author: Information Systems Laboratory, Stanford University, Stanford CA 94305. Phone (415) 723-1538. Fax (415) 723-8473. E-mail: hassibi@rascals.stanford.edu.

### References

J. A. Ball and J. W. Helton. (1992) Nonlinear $H^\infty$ control theory for stable plants. *Math. Control Signals Systems*, 5:233-261.

B. Hassibi, A. H. Sayed, and T. Kailath. (1994) $H^\infty$ optimality criteria for LMS and backpropagation. To appear in *Advances in Neural Information Processing Systems*, Vol. 6, Morgan-Kaufmann.

S. Haykin. (1991) *Adaptive Filter Theory.* Prentice Hall, Englewood Cliffs, NJ.

P. P. Khargonekar and K. M. Nagpal. (1991) Filtering and smoothing in an $H^\infty$ setting. *IEEE Trans. on Automatic Control*, AC-36:831-847.

D. E. Rumelhart, J. L. McClelland and the PDP Research Group. (1986) *Parallel distributed processing : explorations in the microstructure of cognition.* Cambridge, Mass. : MIT Press.

B. Widrow and M. E. Hoff, Jr. (1960) Adaptive switching circuits. *IRE WESCON Conv. Rec.*, Pt.4:96-104.

G. Zames. (1981) Feedback optimal sensitivity: model preference transformation, multiplicative seminorms and approximate inverses. *IEEE Trans. on Automatic Control*, AC-26:301-320.